# Neurophysiological Evidence of Cooperative Mechanisms for Stereo Computation

**Jason M. Samonds**
Center for the Neural Basis
of Cognition (CNBC)
Carnegie Mellon University
Pittsburgh, PA 15213
*samondjm@cnbc.cmu.edu*

**Brian R. Potetz**
CNBC and Computer
Science Department
Carnegie Mellon University
Pittsburgh, PA 15213
*bpotetz@cs.cmu.edu*

**Tai Sing Lee**
CNBC and Computer
Science Department
Carnegie Mellon University
Pittsburgh, PA 15213
*tai@cnbc.cmu.edu*

## Abstract

Although there has been substantial progress in understanding the neuro-physiological mechanisms of stereopsis, how neurons interact in a network during stereo computation remains unclear. Computational models on stereopsis suggest local competition and long-range cooperation are important for resolving ambiguity during stereo matching. To test these predictions, we simultaneously recorded from multiple neurons in V1 of awake, behaving macaques while presenting surfaces of different depths rendered in dynamic random dot stereograms. We found that the interaction between pairs of neurons was a function of similarity in receptive fields, as well as of the input stimulus. Neurons coding the same depth experienced common inhibition early in their responses for stimuli presented at their non-preferred disparities. They experienced mutual facilitation later in their responses for stimulation at their preferred disparity. These findings are consistent with a local competition mechanism that first removes gross mismatches, and a global cooperative mechanism that further refines depth estimates.

## 1 Introduction

The human visual system is able to extract three-dimensional (3D) structures in random noise stereograms even when such images evoke no perceptible patterns when viewed monocularly [1]. Bela Julesz proposed that this is accomplished by a stereopsis mechanism that detects correlated shifts in 2D noise patterns between the two eyes. He also suggested that this mechanism likely involves cooperative neural processing early in the visual system. Marr and Poggio formalized the computational constraints for solving stereo matching (Fig. 1a) and devised an algorithm that can discover the underlying 3D structures in a variety of random dot stereogram patterns [2]. Their algorithm was based on two rules: (1) each element or feature is unique (i.e., can be assigned only one disparity) and (2) surfaces of objects are cohesive (i.e., depth changes gradually across space). To describe their algorithm in neurophysiological terms, we can consider neurons in primary visual cortex as simple element or feature detectors. The first rule is implemented by introducing competitive interactions (mutual inhibition) among neurons of different disparity tuning at each location (Fig. 1b, blue solid horizontal or vertical lines), allowing only one disparity to be detected at each location. The second rule is implemented by introducing cooperative interactions (mutual facilitation) among neurons tuned to the same depth (image disparity) across different spatial locations (Fig. 1b, along the red dashed diagonal lines). In other words, a disparity estimate at one location is more likely to be correct if neighboring locations have similar disparity estimates. A dynamic system under such constraints can relax to a stable global disparity map. Here, we present neurophysiological evidence of interactions between disparity-tuned

neurons in the primary visual cortex that is consistent with this general approach. We sampled from a variety of spatially distributed disparity tuned neurons (see *electrodes* Fig. 1b) while displaying DRDS stimuli defined at various disparities (see *stimulus* Fig.1b). We then measured the dynamics of interactions by assessing the temporal evolution of correlation in neural responses.

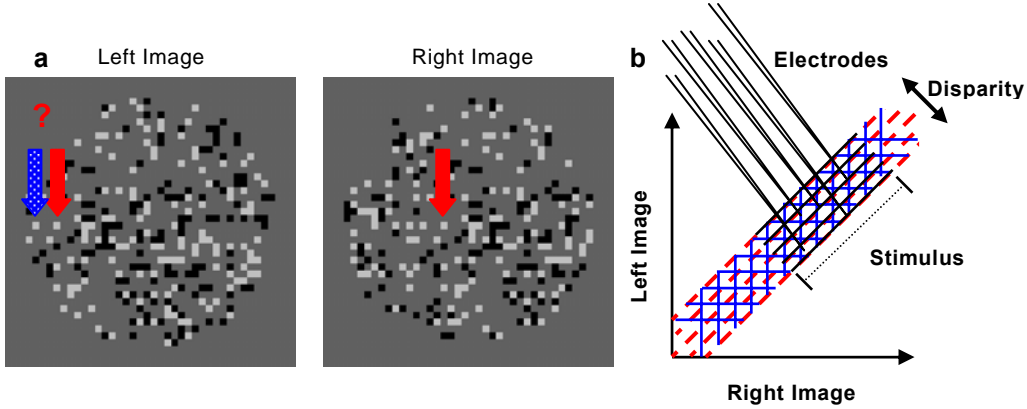

Figure 1: (a) Left and right images of random dot stereogram (right image has been shifted to the right). (b) 1D graphical depiction of competition (blue solid lines) and cooperation (red dashed lines) among disparity-tuned neurons with respect to space as defined by Marr and Poggio's stereo algorithm [2].

## 2 Methods

### 2.1 Recording and stimulation

Recordings were made in V1 of two awake, behaving macaques. We simultaneously recorded from 4-8 electrodes providing data from up to 10 neurons in a single recording session (some electrodes recorded from as many as 3 neurons). We collected data from 112 neurons that provided 224 pairs for cross-correlation analysis. For stimuli, we used 12 Hz dynamic random dot stereograms (DRDS; 25% density black and white pixels on a mean luminance background) presented in a 3.5-degree aperture. Liquid crystal shutter goggles were used to present random dot patterns to each eye separately. Eleven horizontal disparities between the two eyes, ranging from ±0.9 degrees, were tested. Seventy-four neurons (66%) had significant disparity tuning and 99 pairs (44%) were comprised of neurons that *both* had significant disparity tuning (1-way ANOVA, $p<0.05$).

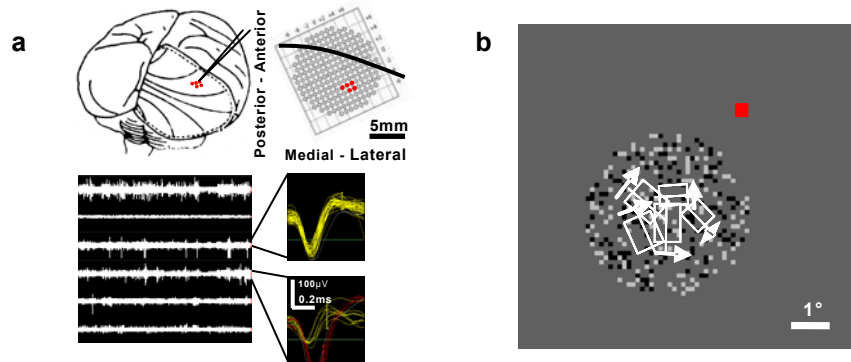

Figure 2: (a) Example recording session from five electrodes in V1. (b) Receptive field (white box—arrow represents direction preference) and random dot stereogram locations for same recording session (small red square is the fixation spot).

## 2.2 Data analysis

Interaction between neurons was described as "effective connectivity" defined by cross-correlation methods [3]. First, the probability of all joint spikes (*x* and *y*) between the two neurons was calculated for all times from stimulus onset ($t_1$ and $t_2$) including all possible lag times ($t_1$ - $t_2$) between the two neurons (2D joint peristimulus time histogram—JPSTH). Next, the cross-product of each neuron's PSTH (joint probabilities expected from chance) was subtracted from the JPSTH; this difference is referred to as the cross-covariance histogram. Finally, the cross-covariance histogram was normalized by the geometric mean of the auto-covariance histograms:

$$C_{x,y}(t_1,t_2) = \frac{\langle x(t_1)y(t_2)\rangle - \langle x(t_1)\rangle\langle y(t_2)\rangle}{\sqrt{\left(\langle x(t_1)x(t_1)\rangle - \langle x(t_1)\rangle\langle x(t_1)\rangle\right)\left(\langle y(t_2)y(t_2)\rangle - \langle y(t_2)\rangle\langle y(t_2)\rangle\right)}} \tag{1}$$

This normalized cross-covariance histogram is a 2D matrix of Pearson's correlation coefficients between the two neurons where the axes represent time from stimulus onset (Figure 3). The principal diagonal also represents time from stimulus onset for correlation and the opposite diagonal represents lag time between the two neurons. We derived three measurements from this matrix to describe the "effective connectivity" between neuron pairs. Using bootstrapped samples of stimulus trials, we estimated 95% confidence intervals for these three measurements [4]. We first integrated along the principal diagonal to produce correlation versus lag time (i.e., the traditional cross-correlation histogram—CCH). We used CCHs to find significant correlation at or near 0 ms lag times (suggesting synaptic connectivity between the neurons). Second, we integrated under the half-height full bandwidth of significant correlation peaks to quantify effective connectivity. Figure 4 shows the population average of normalized CCHs (*n* = 27) and 95% confidence intervals. Finally, we repeated this integration along the principal diagonal to obtain the temporal evolution of effective connectivity (computed with a running 100 ms window).

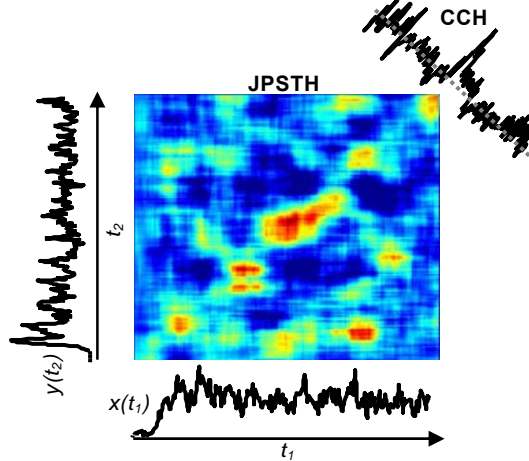

Figure 3: Example normalized cross-covariance histogram.

In computing effective connectivity with Equation 1, we assume trial-to-trial stationarity. If this is not true (e.g., due to difference in attentional effort in different trials), correlation peaks can emerge that are not due to effective connectivity [5]. We applied a correction to equation 1 [5,6] based on the average firing rate for each trial. However, no significant difference in correlation peaks was observed. In addition, changes in DRDS properties other than disparity did not cause significant changes to correlation peak properties. Finally, alternative cross-correlation methods (CCG) [7] using responses to the same exact random dot pattern to predict correlation expected from chance, again, lead to no significant difference in correlation peak properties. These observations justify our assumption that the effective connectivity computed in our case does not arise due to trial-to-trial non-stationarity.

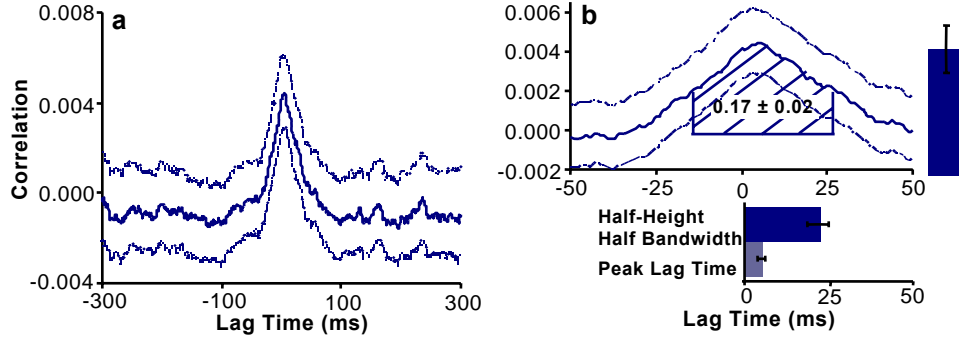

Figure 4: (a) Population average CCH for 27 neuron pairs with a significant correlation peak. (b) Same as (a), but zoomed into ±50 ms lag times with statistics of peak properties (mean ± s.e.m.).

## 3 Interaction depends on tuning properties

The primary indicator of whether or not a neuron pair had a significant correlation peak at or near a 0 ms lag time, for this class of stimuli, was similarity in disparity tuning between the two neurons. Neuron pairs with significant correlation peaks ($n = 27$; 27%) tended to have more similar disparity peaks, bandwidths, and frequencies (determined from fitted Gabor functions) than neuron pairs that did not have significant correlation peaks. We quantified similarity in tuning using the similarity index (*SI*), which is Pearson's product-moment correlation [8]:

$$ SI = \frac{\sum_{i=1}^{n}(x_i - \overline{x})(y_i - \overline{y})}{\left(\sqrt{\sum_{i=1}^{n}(x_i - \overline{x})^2}\right)\left(\sqrt{\sum_{i=1}^{n}(y_i - \overline{y})^2}\right)} \tag{2} $$

where $i$ is each point on the disparity tuning curve, $x$ and $y$ are the firing rates at each point for each neuron, and $\overline{x}$ and $\overline{y}$ are the mean firing rates across the tuning curve. Figure 5a and 5b clearly show that both the probability of correlation and strength in correlation increase with greater *SI* ($n = 27$ pairs). This relationship is limited to long-range interactions among neurons because our electrodes were all at least 1 mm apart. This suggests they are likely mediated by the well known long-range intracortical connections in V1 that link neurons of similar orientation across space [9]. Our results suggest that these connections might also be shared to link similar disparity neurons together. Because connectivity also depended on orientation (Figure 5c), V1 connectivity among neurons appears to depend on similarity across multiple cue dimensions.

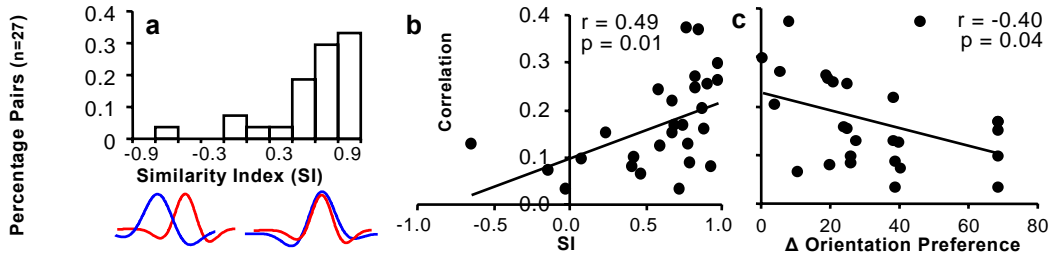

Figure 5: (a) Likelihood of significant correlation peak with respect to similar disparity tuning. (b) Strength of correlation increases with similarity. (c) Correlation is also more likely if orientation preference is similar.

From the 12 pairs of neurons recorded on a single electrode, correlation was observed among neuron pairs with very similar disparity tuning as well as among neurons with nearly opposite disparity tuning (see also [8]). This suggests that antagonistic disparity-tuned neurons tend to spatially coexist, and their interactions are likely competitive.

## 4   Interaction is stimulus-dependent

The interaction between pairs of neurons was not simply a function of the similarity between their receptive field properties but was also a function of the input stimuli (or stimulus disparity in our case). The effective connectivity was significantly modulated (1-way ANOVA, $p < 0.05$) by the stimulus disparity for 25 out of the 27 pairs. We are not suggesting synaptic connections physically change, but rather that the effectiveness of those connections can change depending on the spiking activity and therefore the stimulus input. For neuron pairs with similar disparity tuning, the strongest correlation was observed at their shared preferred disparity, i.e. the peak of the disparity tuning curves based on firing rate (as shown in Figure 6). This suggests facilitation is strongest when a frontal parallel plane activated these neurons simultaneously at their preferred depth. As the stimulus plane moved away from this depth, the effective connectivity between the neurons became weaker. This was observed in 10 pairs (e.g., Figure 6c). For the other 17 pairs (e.g., Figure 6d), the correlation or effective connectivity was again strongest at the neuron pair's shared preferred disparity. However, these pairs in addition exhibited secondary correlation peaks for disparity stimuli that produced the lowest firing rates (even below the baseline for DRDSs).

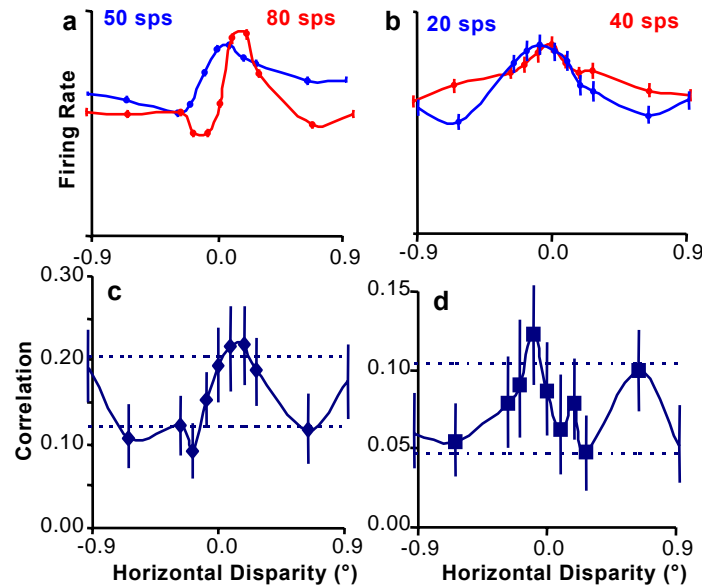

Figure 6: Top row are disparity tuning curves based on firing rates (mean ± s.e.m.). Bottom row are disparity tuning curves based on correlation for the corresponding pairs of neurons in the top row. Error bars are 95% confidence intervals and dashed lines represent 95% confidence of the mean correlation.

Cross-correlation peaks are interpreted as a result of effective circuits that may represent any combination of a variety of synaptic connections that may have a bias in direction (one neuron drives the other) or may not have a bias in direction (zero lag time; both neurons receive a common drive) [10]. As correlation peaks become broader, as in our case (mean = 42 ms), this interpretation becomes more ambiguous (more possible circuits). The broader positive correlation peaks can even be caused by common inhibitory circuitry. One way to potentially disambiguate our interpretations is to consider firing rate behavior. The positive correlation measured at the preferred disparity suggests that the interaction was likely facilitatory in nature based on the increased firing of the neurons. The positive correlation measured at the disparity where both neurons' firing rates were depressed, i.e. at the valley of the firing-

rate based disparity tuning curves, suggests that the correlation likely arose from common inhibition (presumably from neurons that preferred that disparity).

## 5    Temporal dynamics of interaction

We can compare the temporal dynamics of the correlation with the temporal dynamics of the firing rate of the neurons to gain more insight into the possible underlying circuitry. We computed the correlation every 1 msec over a 100 ms running window, and found that the correlation peak at the preferred disparity (based on firing rates) occurred at a later time (250-350 ms post-stimulus onset) than the correlation peaks at the non-preferred disparity (100-200 ms). Figure 7 illustrates the temporal dynamics of correlation for the example neuron pair shown in Figure 6b and 6d. The distinct interval in which correlation emerged at the preferred and the non-preferred disparities was consistently observed for all 27 pairs of neurons. Even for the example shown in Figure 6c, there were peaks in correlation in the early part of the response at the most non-preferred disparities. The timing of these two phases of correlation was also rather consistent over the population of pairs.

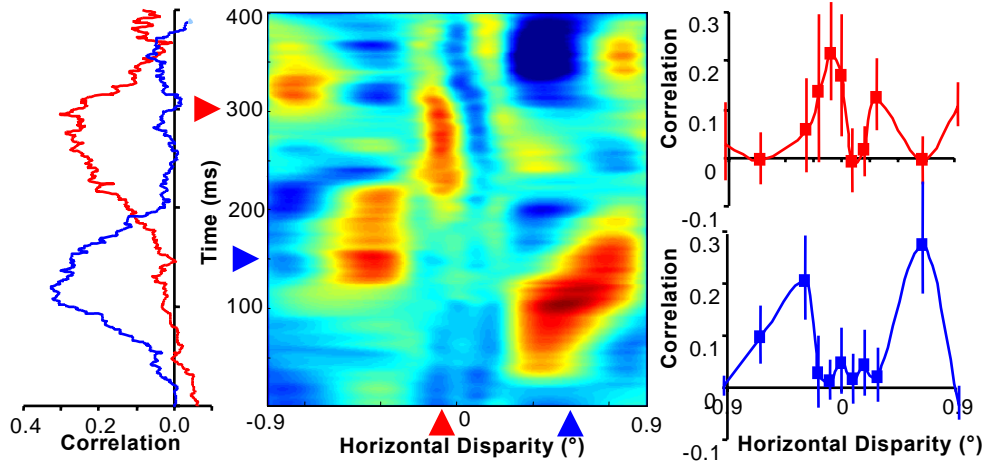

Figure 7: Temporal dynamics of correlation for example neuron pair shown in Figure 6, *right*. From *left* to *right*: Correlation versus time for preferred (red) and non-preferred (blue) disparities. Contour map of correlation versus time and disparity. Disparity tuning based on correlation for the early (blue) and late (red) portion of the response (95% confidence intervals). Correlation was calculated every 1 ms over 100 ms windows.

By examining the interplay between firing rate and correlation, we were able to gain even greater insight about the interactions among neuron pairs. To summarize this interplay across our population, we compared the temporal evolution of the correlation at three distinct disparities with the temporal evolution of the firing rates at the same disparities (also smoothed with 100 ms time windows). The first disparity, the preferred disparity *A*, is where we measured the strongest correlation and was at or near the highest firing rate measured in individual neurons (see Figure 8, left). The second important disparity, the most non-preferred disparity *C*, was where we measured secondary correlation peaks and coincided with the lowest firing rates observed in individual neurons. Lastly, we looked at a disparity *B* that was in between disparities *A* and *C*.

Figure 8 shows that neurons responded better to their preferred disparity over other disparities very early, resulting in immediate moderate firing rate-based disparity tuning. Then shortly after (100 ms), a correlation peak emerges at the least preferred disparity *C*. This coincides with suppression of firing rate for all disparities (Figure 8, *blue dashed line*). However, the suppression in firing rate is much stronger for *C* where the firing rate diverges downward from the firing rates for *A* and *B* sharpening the disparity tuning (Figure 8, *blue arrow*; see also [11]). The strong correlation coupled with the decrease in firing suggests strong common inhibition.

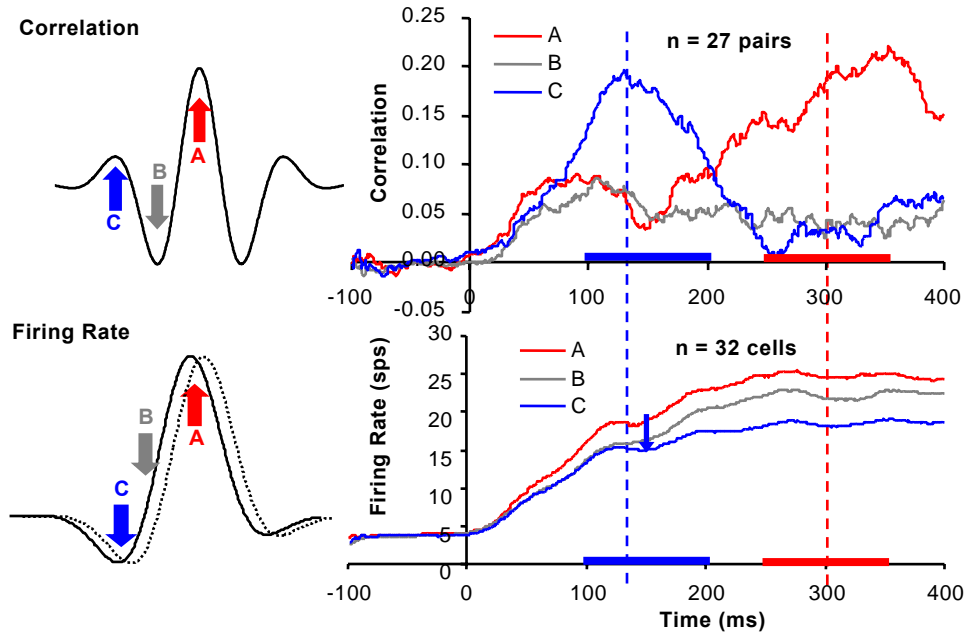

Figure 8: Population average of normalized correlation versus time (*top*) for three disparities shown on the *left*. Population average of normalized PSTHs for same three disparities (*bottom*). Both correlation and firing rates were calculated every 1 ms over 100 ms windows.

Once the correlation peak at *C* subsided (200 ms), the correlation increased for *A* (*red dashed line*). When the correlation for *A* peaked, the correlation decreased for *B* and *C*, leading to very sharp correlation-based disparity tuning (see also Figure 7). This correlation-based tuning can facilitate depth estimates by changing how effectively these signals are integrated downstream as a function of disparity [12].

Our interpretation is that the initial firing rate bias leads to antagonistic disparity-tuned neurons generating common inhibition that suppresses firing at non-preferred disparities, removing potential mismatches. The immediacy suggests that mutual inhibition was local, which is consistent with our observation that many opposing disparity-tuned neurons spatially coexisted. The slower correlation peak at the preferred disparity *A* is indicative of mutual facilitation that occurred when the depth estimates of spatially distinct neurons matched. This facilitation leads to a more precise estimate of depth.

# 6  Discussion and conclusions

The findings from this study provide support to Julesz's proposal that cooperative and competitive mechanisms in primary visual cortex are utilized for estimating global depth in random dot stereograms [1], which was later described formally by Marr and Poggio [2]. More recent cooperative stereo computation models allow excitatory interaction between neurons of different disparities separated by long distance. This is used to accommodate the computation of slanted surfaces [13,14]. In this experiment, we only tested frontal parallel planes, thus, we cannot answer whether or not effective connections and facilitation exist between neurons with larger disparity differences over long distances. This will require further experiments using planes with disparity gradients.

The observation that initial correlation peaks occurred at disparities that evoked the lowest firing rates in neurons, suggests that correlation peaks emerged from common inhibition for non-preferred disparities. The observation that later correlation occurred at disparities that evoked the highest firing rates suggests that neurons were mutually exciting each other at their preferred disparity. Our neurophysiological data reveal interesting dynamics between network-based (effective connectivity) and firing rate-based encoding of depth estimates. The observation that inhibition precedes facilitation suggests that competition is local (re-

calling neurons at the same electrode tend to have opposite disparity tuning) and cooperation is more global (mediated through long-range connectivity). Local competition between neurons encoding different depths is consistent with the uniqueness principle of Marr and Poggio's algorithm [2]. In addition, cooperation among neurons encoding the same depth across space was predicted by the second rule of their algorithm: matter is cohesive. These two interactions are robust at removing potential ambiguity during stereo matching and depth inference.

Previous neurophysiological data had suggested that intracortical connectivity in primary visual cortex underlies competitive [15] and cooperative [16] mechanisms for improving estimates of orientation. Our data suggests similar circuitry might play a role also in stereo matching [17]. However, this study is distinct in that it provides detailed empirical support for computational algorithms for solving stereo matching. It thus highlights the importance of computational algorithms in generating hypotheses to guide future neurophysiological studies.

## Acknowledgments

We thank George Gerstein and Jeff Keating for JPSTH software. Supported by NIMH IBSC MH64445 and NSF CISE IIS-0413211 grants.

## References

[1] Julesz, B. (1971) *Foundations of cyclopean perception.* Chicago: University of Chicago Press.

[2] Marr, D. & Poggio, T. (1976) Cooperative computation of stereo disparity. *Science* **194**(4262):283-287.

[3] Aertsen, A.M., Gerstein, G.L., Habib, M.K. & Palm, G. (1989) Dynamics of neuronal firing correlation: modulation of "effective connectivity". *Journal of Neurophysiology* **61**(5):900-917.

[4] Efron, B. & Tibshirani, R. (1993) *An Introduction to the Bootstrap.* New York: Chapman & Hall.

[5] Brody, C.D. (1999) Correlations without synchrony. *Neural Computation* **11**(7):1537-1551.

[6] Gerstein, G.L. & Kirkland, K.L. (2001) Neural assemblies: technical issues, analysis, and modeling. *Neural Networks* **14**(6-7):589-598.

[7] Kohn, A. & Smith, M.A. (2005) Stimulus dependence of neuronal correlation in primary visual cortex of the macaque. *Journal of Neuroscience* **25**(14):3661-3673.

[8] Menz, M. & Freeman, R.D. (2004) Functional connectivity of disparity-tuned neurons in the visual cortex. *Journal of Neurophysiology* **91**(4):1794-1807.

[9] Gilbert, C.D. & Wiesel, T.N. (1989) Columnar specificity of intrinsic horizontal and corticocortical connections in cat visual cortex. *Journal of Neuroscience* **9**(7):2432-2442.

[10] Moore, G.P., Segundo, J.P., Perkel, D.H. & Levitan, H. (1970) Statistical signs of synaptic interaction in neurons. *Biophysics Journal* **10**(9):876-900.

[11] Menz, M. & Freeman, R.D. (2003) Stereoscopic depth processing in the visual cortex: a coarse-to-fine mechanism. *Nature Neuroscience* **6**(1):59-65.

[12] Bruno, R.M. & Sakmann, B. (2006) Cortex is driven by weak but synchronously active thalamocortical synapses. *Science* **312**(5780):1622-1627.

[13] Prazdny, K. (1985) Detection of binocular disparities. *Biological Cybernetics* **52**(2):93-99.

[14] Pollard, S.B., Mayhew, J.E., & Frisby, J.P. (1985) PMF: a stereo correspondence algorithm using a disparity gradient limit. *Perception* **14**(4):449-470.

[15] Ringach, D.L., Hawken, M.J. & Shapley, R. (1997) Dynamics of orientation tuning in macaque primary visual cortex. *Nature* **387**(6630):281-284.

[16] Samonds, J.M., Allison, J.D., Brown, H.A. & Bonds, A.B. (2004) Cooperative synchronized assemblies enhance orientation discrimination. *Proceedings of the National Academy of Sciences USA* **101**(17):6722-6727.

[17] Ben-Shahar, O., Huggins, P.S., Izo, T. & Zucker, S.W. (2003) Cortical connections and early visual function: intra- and inter-columnar processing. *Journal of Physiology (Paris)* **97**(2-3):191-208.
